# Colored Maximum Variance Unfolding

**Le Song[†], Alex Smola[†], Karsten Borgwardt[‡] and Arthur Gretton[*]**
[†]National ICT Australia, Canberra, Australia
[‡]University of Cambridge, Cambridge, United Kingdom
[*]MPI for Biological Cybernetics, Tübingen, Germany
{le.song,alex.smola}@nicta.com.au
kmb51@eng.cam.ac.uk,arthur.gretton@tuebingen.mpg.de

## Abstract

Maximum variance unfolding (MVU) is an effective heuristic for dimensionality reduction. It produces a low-dimensional representation of the data by maximizing the variance of their embeddings while preserving the local distances of the original data. We show that MVU also optimizes a statistical dependence measure which aims to retain the identity of individual observations under the distance-preserving constraints. This general view allows us to design "colored" variants of MVU, which produce low-dimensional representations for a given task, e.g. subject to class labels or other side information.

## 1 Introduction

In recent years maximum variance unfolding (MVU), introduced by Saul et al. [1], has gained popularity as a method for dimensionality reduction. This method is based on a simple heuristic: maximizing the overall variance of the embedding while preserving the local distances between neighboring observations. Sun et al. [2] show that there is a dual connection between MVU and the goal of finding a fast mixing Markov chain. This connection is intriguing. However, it offers limited insight as to why MVU can be used for data representation.

This paper provides a statistical interpretation of MVU. We show that the algorithm attempts to extract features from the data which simultaneously preserve the identity of individual observations *and* their local distance structure. Our reasoning relies on a dependence measure between sets of observations, the Hilbert-Schmidt Independence Criterion (HSIC) [3].

Relaxing the requirement of retaining maximal information about individual observations, we are able to obtain "colored" MVU. Unlike traditional MVU which takes only *one* source of information into account, "colored" MVU allows us to integrate *two* sources of information into a single embedding. That is, we are able to find an embedding that leverages between two goals:

- preserve the local distance structure according to the *first* source of information (the data);
- and maximally align with the *second* sources of information (side information).

Note that not all features inherent in the data are interesting for an ulterior objective. For instance, if we want to retain a reduced representation of the data for later classification, then only those discriminative features will be relevant. "Colored" MVU achieves the goal of elucidating primarily relevant features by aligning the embedding to the objective provided in the side information. Some examples illustrate this situation in more details:

- Given a-bag-of-pixels representation of images (the data), such as USPS digits, find an embedding which reflects the categories of the images (side information).
- Given a vector space representation of texts on the web (the data), such as newsgroups, find an embedding which reflects a hierarchy of the topics (side information).

- Given a TF/IDF representation of documents (the data), such as NIPS papers, find an embedding which reflects co-authorship relations between the documents (side information).

There is a strong motivation for *not* simply merging the two sources of information into a single distance metric: Firstly, the data and the side information may be heterogenous. It is unclear how to combine them into a single distance metric; Secondly, the side information may appear in the form of similarity rather than distance. For instance, co-authorship relations is a similarity between documents (if two papers share more authors, they tends to be more similar), but it does not induce a distance between the documents (if two papers share no authors, we cannot assert they are far apart). Thirdly, at test time (i.e. when inserting a new observation into an existing embedding) only one source of information might be available, i.e. the side information is missing.

## 2 Maximum Variance Unfolding

We begin by giving a brief overview of MVU and its projection variants, as proposed in [1]. Given a set of $m$ observations $Z = \{z_1, \ldots, z_m\} \subseteq \mathcal{Z}$ and a distance metric $d : \mathcal{Z} \times \mathcal{Z} \to [0, \infty)$ find an inner product matrix (kernel matrix) $\mathbf{K} \in \mathbb{R}^{m \times m}$ with $\mathbf{K} \succeq 0$ such that

1. The distances are preserved, i.e. $\mathbf{K}_{ii} + \mathbf{K}_{jj} - 2\mathbf{K}_{ij} = d_{ij}^2$ for all $(i, j)$ pairs which are sufficiently close to each other, such as the $n$ nearest neighbors of each observation. We denote this set by $\mathcal{N}$. We will also use $\mathcal{N}$ to denote the graph formed by having these $(i, j)$ pairs as edges.
2. The embedded data is centered, i.e. $\mathbf{K}\mathbf{1} = \mathbf{0}$ (where $\mathbf{1} = (1, \ldots, 1)^\top$ and $\mathbf{0} = (0, \ldots, 0)^\top$).
3. The trace of $\mathbf{K}$ is maximized (the maximum variance unfolding part).

Several variants of this algorithm, including a large scale variant [4] have been proposed. By and large the optimization problem looks as follows:

$$\underset{\mathbf{K} \succeq 0}{\text{maximize}} \ \text{tr} \, \mathbf{K} \text{ subject to } \mathbf{K}\mathbf{1} = \mathbf{0} \text{ and } \mathbf{K}_{ii} + \mathbf{K}_{jj} - 2\mathbf{K}_{ij} = d_{ij}^2 \text{ for all } (i, j) \in \mathcal{N}. \quad (1)$$

Numerous variants of (1) exist, e.g. where the distances are only allow to shrink, where slack variables are added to the objective function to allow approximate distance preserving, or where one uses low-rank expansions of $\mathbf{K}$ to cope with the computational complexity of semidefinite programming.

A major drawback with MVU is that its results necessarily come as somewhat of a surprise. That is, it is never clear before invoking MVU what specific interesting results it might produce. While in hindsight it is easy to find an insightful interpretation of the outcome, it is not a-priori clear which aspect of the data the representation might emphasize. A second drawback is that while in general generating brilliant results, its statistical origins are somewhat more obscure. We aim to address these problems by means of the Hilbert-Schmidt Independence Criterion.

## 3 Hilbert-Schmidt Independence Criterion

Let sets of observations $X$ and $Y$ be drawn jointly from some distribution $\Pr_{xy}$. The Hilbert-Schmidt Independence Criterion (HSIC) [3] measures the dependence between two random variables, $x$ and $y$, by computing the square of the norm of the cross-covariance operator over the domain $\mathcal{X} \times \mathcal{Y}$ in Hilbert Space. It can be shown, provided the Hilbert Space is universal, that this norm vanishes if and only if $x$ and $y$ are independent. A large value suggests strong dependence with respect to the choice of kernels.

Let $\mathcal{F}$ and $\mathcal{G}$ be the reproducing kernel Hilbert Spaces (RKHS) on $\mathcal{X}$ and $\mathcal{Y}$ with associated kernels $k : \mathcal{X} \times \mathcal{X} \to \mathbb{R}$ and $l : \mathcal{Y} \times \mathcal{Y} \to \mathbb{R}$ respectively. The cross-covariance operator $\mathcal{C}_{xy} : \mathcal{G} \mapsto \mathcal{F}$ is defined as [5]

$$\mathcal{C}_{xy} = \mathbb{E}_{xy} \left[ (k(x, \cdot) - \mu_x)(l(y, \cdot) - \mu_y) \right], \quad (2)$$

where $\mu_x = \mathbb{E}[k(x, \cdot)]$ and $\mu_y = \mathbb{E}[l(y, \cdot)]$. HSIC is then defined as the square of the Hilbert-Schmidt norm of $\mathcal{C}_{xy}$, that is $\text{HSIC}(\mathcal{F}, \mathcal{G}, \Pr_{xy}) := \|\mathcal{C}_{xy}\|_{\text{HS}}^2$. In term of kernels HSIC is

$$\mathbb{E}_{xx'yy'}[k(x, x')l(y, y')] + \mathbb{E}_{xx'}[k(x, x')]\mathbb{E}_{yy'}[l(y, y')] - 2\mathbb{E}_{xy}[\mathbb{E}_{x'}[k(x, x')]\mathbb{E}_{y'}[l(y, y')]]. \quad (3)$$

Given the samples $(X, Y) = \{(x_1, y_1), \ldots, (x_m, y_m)\}$ of size $m$ drawn from the joint distribution, $\Pr_{xy}$, an empirical estimate of HSIC is [3]

$$\text{HSIC}(\mathcal{F}, \mathcal{G}, Z) = (m-1)^{-2} \operatorname{tr} \mathbf{HKHL}, \qquad (4)$$

where $\mathbf{K}, \mathbf{L} \in \mathbb{R}^{m \times m}$ are the kernel matrices for the data and the labels respectively, and $\mathbf{H}_{ij} = \delta_{ij} - m^{-1}$ centers the data and the labels in the feature space. (For convenience, we will drop the normalization and use $\operatorname{tr} \mathbf{HKHL}$ as HSIC.)

HSIC has been used to measure independence between random variables [3], to select features or to cluster data (see the Appendix for further details). Here we use it in a different way:

> We try to construct a kernel matrix $\mathbf{K}$ for the dimension-reduced data $X$ which preserves the local distance structure of the original data $Z$, such that $X$ is maximally dependent on the side information $Y$ as seen from its kernel matrix $\mathbf{L}$.

HSIC has several advantages as a dependence criterion. First, it satisfies concentration of measure conditions [3]: for random draws of observation from $\Pr_{xy}$, HSIC provides values which are very similar. This is desirable, as we want our metric embedding to be robust to small changes. Second, HSIC is easy to compute, since only the kernel matrices are required and no density estimation is needed. The freedom of choosing a kernel for $\mathbf{L}$ allows us to incorporate prior knowledge into the dependence estimation process. The consequence is that we are able to incorporate various side information by simply choosing an appropriate kernel for $Y$.

## 4 Colored Maximum Variance Unfolding

We state the algorithmic modification first and subsequently we explain why this is reasonable: the key idea is to replace $\operatorname{tr} \mathbf{K}$ in (1) by $\operatorname{tr} \mathbf{KL}$, where $\mathbf{L}$ is the covariance matrix of the domain (side information) with respect to which we would like to extract features. For instance, in the case of NIPS papers which happen to have author information, $\mathbf{L}$ would be the kernel matrix arising from coauthorship and $d(z, z')$ would be the Euclidean distance between the vector space representations of the documents. Key to our reasoning is the following lemma:

**Lemma 1** *Denote by $\mathbf{L}$ a positive semidefinite matrix in $\mathbb{R}^{m \times m}$ and let $\mathbf{H} \in \mathbb{R}^{m \times m}$ be defined as $\mathbf{H}_{ij} = \delta_{ij} - m^{-1}$. Then the following two optimization problems are equivalent:*

$$\underset{\mathbf{K}}{\operatorname{maximize}} \operatorname{tr} \mathbf{HKHL} \ \textit{subject to } \mathbf{K} \succeq 0 \textit{ and constraints on } \mathbf{K}_{ii} + \mathbf{K}_{jj} - 2\mathbf{K}_{ij}. \qquad (5a)$$

$$\underset{\mathbf{K}}{\operatorname{maximize}} \operatorname{tr} \mathbf{KL} \ \textit{subject to } \mathbf{K} \succeq 0 \textit{ and constraints on } \mathbf{K}_{ii} + \mathbf{K}_{jj} - 2\mathbf{K}_{ij} \textit{ and } \mathbf{K1} = \mathbf{0}. \qquad (5b)$$

*Any solution of (5b) solves (5a) and any solution of (5a) solves (5b) after centering $\mathbf{K} \leftarrow \mathbf{HKH}$.*

**Proof** Denote by $\mathbf{K}_a$ and $\mathbf{K}_b$ the solutions of (5a) and (5b) respectively. $\mathbf{K}_b$ is feasible for (5a) and $\operatorname{tr} \mathbf{K}_b \mathbf{L} = \operatorname{tr} \mathbf{HK}_b \mathbf{HL}$. This implies that $\operatorname{tr} \mathbf{HK}_a \mathbf{HL} \geq \operatorname{tr} \mathbf{HK}_b \mathbf{HL}$. Vice versa $\mathbf{HK}_a \mathbf{H}$ is feasible for (5b). Moreover $\operatorname{tr} \mathbf{HK}_a \mathbf{HL} \leq \operatorname{tr} \mathbf{K}_b \mathbf{L}$ by requirement on the optimality of $\mathbf{K}_b$. Combining both inequalities shows that $\operatorname{tr} \mathbf{HK}_a \mathbf{HL} = \operatorname{tr} \mathbf{K}_b \mathbf{L}$, hence both solutions are equivalent. ∎

This means that the centering imposed in MVU via constraints is equivalent to the centering in HSIC by means of the dependence measure $\operatorname{tr} \mathbf{HKHL}$ itself. In other words, MVU equivalently maximizes $\operatorname{tr} \mathbf{HKHI}$, i.e. the dependence between $\mathbf{K}$ and the identity matrix $\mathbf{I}$ which corresponds to retain maximal diversity between observations via $\mathbf{L}_{ij} = \delta_{ij}$. This suggests the following colored version of MVU:

$$\underset{\mathbf{K}}{\operatorname{maximize}} \operatorname{tr} \mathbf{HKHL} \ \textit{subject to } \mathbf{K} \succeq 0 \text{ and } \mathbf{K}_{ii} + \mathbf{K}_{jj} - 2\mathbf{K}_{ij} = d_{ij}^2 \text{ for all } (i,j) \in \mathcal{N}. \qquad (6)$$

Using (6) we see that we are now extracting a Euclidean embedding which maximally depends on the coloring matrix $\mathbf{L}$ (for the side information) while preserving local distance structure. A second advantage of (6) is that whenever we restrict $\mathbf{K}$ further, e.g. by only allowing for $\mathbf{K}$ be part of a linear subspace formed by the principal vectors in some space, (6) remains feasible, whereas the (constrained) MVU formulation may become infeasible (i.e. $\mathbf{K1} = \mathbf{0}$ may not be satisfied).

## 5 Dual Problem

To gain further insight into the structure of the solution of (6) we derive its dual problem. Our approach uses the results from [2]. First we define matrices $\mathbf{E}^{ij} \in \mathbb{R}^{m \times m}$ for each edge $(i,j) \in \mathcal{N}$, such that it has only four nonzero entries $\mathbf{E}^{ij}_{ii} = \mathbf{E}^{ij}_{jj} = 1$ and $\mathbf{E}^{ij}_{ij} = \mathbf{E}^{ij}_{ji} = -1$. Then the distance preserving constraint can be written as $\operatorname{tr} \mathbf{K} \mathbf{E}^{ij} = d^2_{ij}$. Thus we have the following Lagrangian:

$$
\begin{aligned}
\mathcal{L} &= \operatorname{tr} \mathbf{K} \mathbf{H} \mathbf{L} \mathbf{H} + \operatorname{tr} \mathbf{K} \mathbf{Z} - \sum_{(i,j) \in \mathcal{N}} w_{ij}(\operatorname{tr} \mathbf{K} \mathbf{E}^{ij} - d^2_{ij}) \\
&= \operatorname{tr} \mathbf{K}(\mathbf{H} \mathbf{L} \mathbf{H} + \mathbf{Z} - \sum_{(i,j) \in \mathcal{N}} w_{ij} \mathbf{E}^{ij}) + \sum_{(i,j) \in \mathcal{N}} w_{ij} d^2_{ij} \text{ where } \mathbf{Z} \succeq 0 \text{ and } w_{ij} \geq 0.
\end{aligned}
\tag{7}
$$

Setting the derivative of $\mathcal{L}$ with respect to $\mathbf{K}$ to zero, yields $\mathbf{H} \mathbf{L} \mathbf{H} + \mathbf{Z} - \sum_{(i,j) \in \mathcal{N}} w_{ij} \mathbf{E}^{ij} = 0$. Plugging this condition into (7) gives us the dual problem.

$$
\underset{w_{ij}}{\operatorname{minimize}} \sum_{(i,j) \in \mathcal{N}} w_{ij} d^2_{ij} \text{ subject to } \mathbf{G}(w) \succeq \mathbf{H} \mathbf{L} \mathbf{H} \text{ where } \mathbf{G}(w) = \sum_{(i,j) \in \mathcal{N}} w_{ij} \mathbf{E}^{ij}.
\tag{8}
$$

Note that $\mathbf{G}(w)$ amounts to the Graph Laplacian of a weighted graph with adjacency matrix given by $w$. The dual constraint $\mathbf{G}(w) \succeq \mathbf{H} \mathbf{L} \mathbf{H}$ effectively requires that the eigen-spectrum of the graph Laplacian is bounded from below by that of $\mathbf{H} \mathbf{L} \mathbf{H}$.

We are interested in the properties of the solution $\mathbf{K}$ of the primal problem, in particular the number of nonzero eigenvalues. Recall that at optimality the Karush-Kuhn-Tucker conditions imply $\operatorname{tr} \mathbf{K} \mathbf{Z} = 0$, i.e. the row space of $\mathbf{K}$ lies in the null space of $\mathbf{Z}$. Thus the rank of $\mathbf{K}$ is upper bounded by the dimension of the null space of $\mathbf{Z}$.

Recall that $\mathbf{Z} = \mathbf{G}(w) - \mathbf{H} \mathbf{L} \mathbf{H} \succeq 0$, and by design $\mathbf{G}(w) \succeq 0$ since it is the graph Laplacian of a weighted graph with edge weights $w_{ij}$. If $\mathbf{G}(w)$ corresponds to a connected graphs, only one eigenvalue of $\mathbf{G}(w)$ vanishes. Hence, the eigenvectors of $\mathbf{Z}$ with zero eigenvalues would correspond to those lying in the image of $\mathbf{H} \mathbf{L} \mathbf{H}$. If $\mathbf{L}$ arises from a label kernel matrix, e.g. for an $n$-class classification problem, then we will only have up to $n$ vanishing eigenvalues in $\mathbf{Z}$. This translates into only up to $n$ nonvanishing eigenvalues in $\mathbf{K}$.

Contrast this observation with plain MVU. In this case $\mathbf{L} = \mathbf{I}$, that is, only one eigenvalue of $\mathbf{H} \mathbf{L} \mathbf{H}$ vanishes. Hence it is likely that $\mathbf{G}(w) - \mathbf{H} \mathbf{L} \mathbf{H}$ will have many vanishing eigenvalues which translates into many nonzero eigenvalues of $\mathbf{K}$. This is corroborated by experiments (Section 7).

## 6 Implementation Details

In practice, instead of requiring the distances to remain unchanged in the embedding we only require them to be preserved approximately [4]. We do so by penalizing the slackness between the original distance and the embedding distance, i.e.

$$
\underset{\mathbf{K}}{\operatorname{maximize}} \ \operatorname{tr} \mathbf{H} \mathbf{K} \mathbf{H} \mathbf{L} - \nu \sum_{(i,j) \in \mathcal{N}} \left( \mathbf{K}_{ii} + \mathbf{K}_{jj} - 2\mathbf{K}_{ij} - d^2_{ij} \right)^2 \text{ subject to } \mathbf{K} \succeq 0
\tag{9}
$$

Here $\nu$ controls the tradeoff between dependence maximization and distance preservation. The semidefinite program usually has a time complexity up to $O(m^6)$. This renders direct implementation of the above problem infeasible for anything but toy problems. To reduce the computation, we approximate $\mathbf{K}$ using an orthonormal set of vectors $\mathbf{V}$ (of size $m \times n$) and a smaller positive definite matrix $\mathbf{A}$ (of size $n \times n$), i.e. $\mathbf{K} = \mathbf{V} \mathbf{A} \mathbf{V}^\top$. Conveniently we choose the number of dimensions $n$ to be much smaller than $m$ ($n \ll m$) such that the resulting semidefinite program with respect to $\mathbf{A}$ becomes tractable (clearly this is an approximation).

To obtain the matrix $\mathbf{V}$ we employ a regularization scheme as proposed in [4]. First, we construct a nearest neighbor graph according to $\mathcal{N}$ (we will also refer to this graph and its adjacency matrix as $\mathcal{N}$). Then we form $\mathbf{V}$ by stacking together the bottom $n$ eigenvectors of the graph Laplacian of the neighborhood graph via $\mathcal{N}$. The key idea is that neighbors in the original space remain neighbors in

the embedding space. As we require them to have similar locations, the bottom eigenvectors of the graph Laplacian provide a set of good bases for functions smoothly varying across the graph.

Subsequent to the semidefinite program we perform local refinement of the embedding via gradient descent. Here the objective is reformulated using an $m \times n$ dimensional vector $\mathbf{X}$, i.e. $\mathbf{K} = \mathbf{X}\mathbf{X}^\top$. The initial value $\mathbf{X}_0$ is obtained using the $n$ leading eigenvectors of the solution of (9).

## 7   Experiments

Ultimately the justification for an algorithm is practical applicability. We demonstrate this based on three datasets: embedding of digits of the USPS database, the Newsgroups 20 dataset containing Usenet articles in text form, and a collection of NIPS papers from 1987 to 1999.[1] We compare "colored" MVU (also called MUHSIC, maximum unfolding via HSIC) to MVU [1] and PCA, highlighting places where MUHSIC produces more meaningful results by incorporating side information. Further details, such as effects of the adjacency matrices and a comparison to Neighborhood Component Analysis [6] are relegated to the appendix due to limitations of space.

For images we use the Euclidean distance between pixel values as the base metric. For text documents, we perform four standard preprocessing steps: (*i*) the words are stemmed using the Porter stemmer; (*ii*) we filter out common but meaningless stopwords; (*iii*) we delete words that appear in less than 3 documents; (*iv*) we represent each document as a vector using the usual TF/IDF (term frequency / inverse document frequency) weighting scheme. As before, the Euclidean distance on those vectors is used to find the nearest neighbors.

As in [4] we construct the nearest neighbor graph by considering the 1% nearest neighbors of each point. Subsequently the adjacency matrix of this graph is symmetrized. The regularization parameter $\nu$ as given in (9) is set to 1 as a default. Moreover, as in [4] we choose 10 dimensions ($n = 10$) to decompose the embedding matrix $\mathbf{K}$. Final visualization is carried out using 2 dimensions. This makes our results very comparable to previous work.

**USPS Digits**   This dataset consists of images of hand written digits of a resolution of $16 \times 16$ pixels. We normalized the data to the range $[-1, 1]$ and used the test set containing 2007 observations. Since it is a digit recognition task, we have $\mathcal{Y} \in [0, \ldots, 9]$. $\mathcal{Y}$ is used to construct the matrix $\mathbf{L}$ by applying the kernel $k(y, y') = \delta_{y,y'}$. This kernel further promotes embedding where images from the same class are grouped tighter. Figure 1 shows the results produced by MUHSIC, MVU and PCA.

The overall properties the embeddings are similar across the three methods ('2' on the left, '1' on the right, '7' on top, and '8' at the bottom). Arguably MUHSIC produces a clearer visualization. For instance, images of '5' are clustered tighter in this case than the other two methods. Furthermore, MUHSIC also results in much better separation between images from different classes. For instance, the overlap between '4' and '6' produce by MVU and PCA are largely reduced by MUHSIC. Similar results also hold for '0' and '5'.

Figure 1 also shows the eigenspectrum of $\mathbf{K}$ produced by different methods. The eigenvalues are sorted in descending order and normalized by the trace of $\mathbf{K}$. Each patch in the color bar represents an eigenvalue. We see that MUHSIC results in 3 significant eigenvalues, MVU results in 10, while PCA produces a grading of many eigenvalues (as can be seen by an almost continuously changing spectrum in the spectral diagram). This confirms our reasoning of Section 5 that the spectrum generated by MUHSIC is likely to be considerably sparser than that of MVU.

**Newsgroups**   This dataset consists of Usenet articles collected from 20 different newsgroups. We use a subset of 2000 documents for our experiments (100 articles from each newsgroup). We remove the headers from the articles before the preprocessing while keeping the subject line. There is a clear hierarchy in the newsgroups. For instance, 5 topics are related to computer science, 3 are related to religion, and 4 are related to recreation. We will use these different topics as side information and apply a delta kernel $k(y, y') = \delta_{y,y'}$ on them. Similar to USPS digits we want to preserve the identity of individual newsgroups. While we did not encode hierarchical information for MVU we recover a meaningful hierarchy among topics, as can be seen in Figure 2.

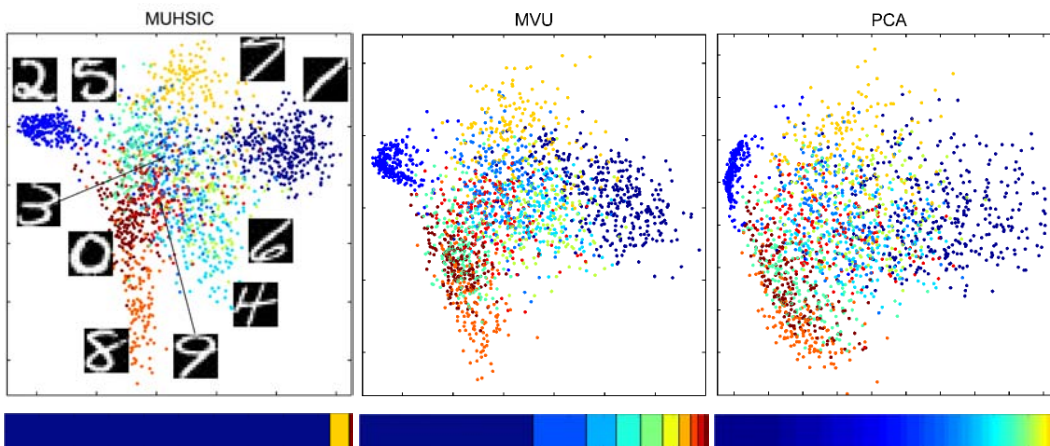

Figure 1: Embedding of 2007 USPS digits produced by MUHSIC, MVU and PCA respectively. Colors of the dots are used to denote digits from different classes. The color bar below each figure shows the eigenspectrum of the learned kernel matrix $\mathbf{K}$.

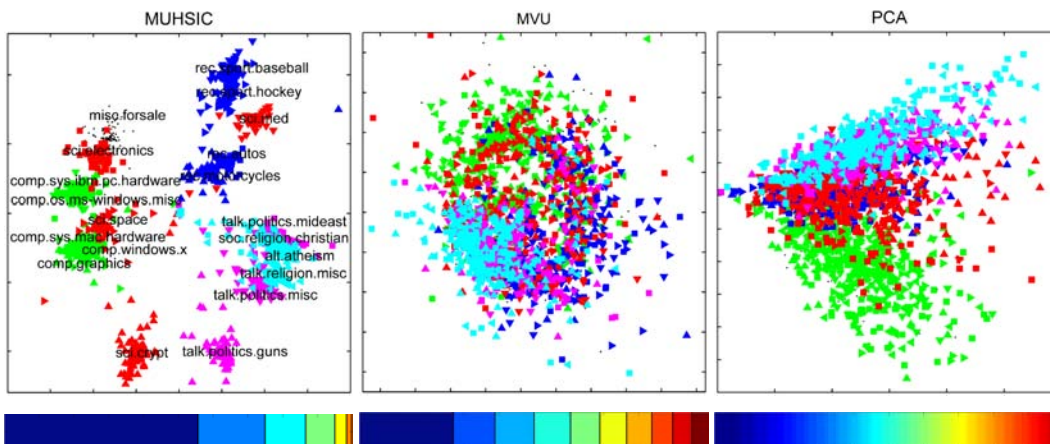

Figure 2: Embedding of 2000 newsgroup articles produced by MUHSIC, MVU and PCA respectively. Colors and shapes of the dots are used to denote articles from different newsgroups. The color bar below each figure shows the eigenspectrum of the learned kernel matrix $\mathbf{K}$.

A distinctive feature of the visualizations is that MUHSIC groups articles from individual topics more tightly than MVU and PCA. Furthermore, the semantic information is also well preserved by MUHSIC. For instance, on the left side of the embedding, all computer science topics are placed adjacent to each other; *comp.sys.ibm.pc.hardware* and *comp.os.ms-windows.misc* are adjacent and well separated from *comp.sys.mac.hardware* and *comp.windows.x* and *comp.graphics*. The latter is meaningful since Apple computers are more popular in graphics (so are X windows based systems for scientific visualization). Likewise we see that on the top we find all recreational topics (with rec.sport.baseball and rec.sport.hockey clearly distinguished from the rec.autos and rec.motorcycles groups). A similar adjacency between *talk.politics.mideast* and *soc.religion.christian* is quite interesting. The layout suggests that the content of *talk.politics.guns* and of *sci.crypt* is quite different from other Usenet discussions.

**NIPS Papers**   We used the 1735 regular NIPS papers from 1987 to 1999. They are scanned from the proceedings and transformed into text files via OCR. The table of contents (TOC) is also available. We parse the TOC and construct a coauthor network from it. Our goal is to embed the papers by taking the coauthor information into account. As kernel $k(y, y')$ we simply use the number of authors shared by two papers. To illustrate this we highlighted some known researchers. Furthermore, we also annotated some papers to show the semantics revealed by the embedding. Figure 3 shows the results produced by MUHSIC, MVU and PCA.

All three methods correctly represent the two major topics of NIPS papers: artificial systems, i.e. machine learning (they are positioned on the left side of the visualization) and natural systems,

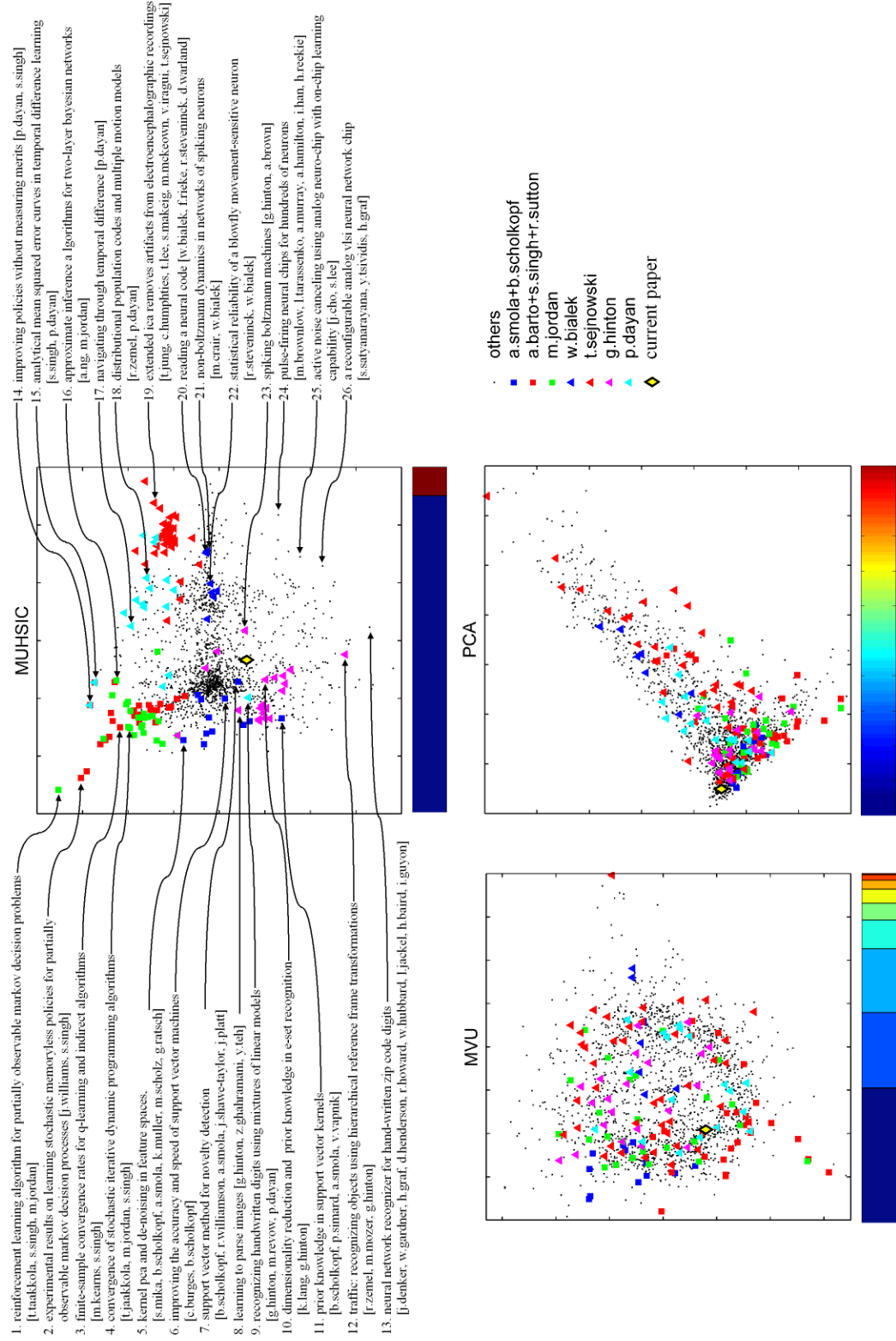

1. reinforcement learning algorithm for partially observable markov decision problems [t.taakkola, s.singh, m.jordan]

2. experimental results on learning stochastic memoryless policies for partially observable markov decision processes [j.williams, s.singh]

3. finite-sample convergence rates for q-learning and indirect algorithms [m.kearns, s.singh]

4. convergence of stochastic iterative dynamic programming algorithms [t.jaakkola, m.jordan, s.singh]

5. kernel pca and de-noising in feature spaces. [s.mika, b.scholkopf, a.smola, k.muller, m.scholz, g.ratsch]

6. improving the accuracy and speed of support vector machines [c.burges, b.scholkopf]

7. support vector method for novelty detection [b.scholkopf, r.williamson, a.smola, j.shawe-taylor, j.platt]

8. learning to parse images [g.hinton, z.ghahramani, y.teh]

9. recognizing handwritten digits using mixtures of linear models [g.hinton, m.revow, p.dayan]

10. dimensionality reduction and prior knowledge in e-set recognition [k.lang, g.hinton]

11. prior knowledge in support vector kernels [b.scholkopf, p.simard, a.smola, v.vapnik]

12. traffic: recognizing objects using hierarchical reference frame transformations [r.zemel, m.mozer, g.hinton]

13. neural network recognizer for hand-written zip code digits [j.denker, w.gardner, h.graf, d.henderson, r.howard, w.hubbard, l.jackel, h.baird, i.guyon]

14. improving policies without measuring merits [p.dayan, s.singh]

15. analytical mean squared error curves in temporal difference learning [s.singh, p.dayan]

16. approximate inference a lgorithms for two-layer bayesian networks [a.ng, m.jordan]

17. navigating through temporal difference [p.dayan]

18. distributional population codes and multiple motion models [r.zemel, p.dayan]

19. extended ica removes artifacts from electroencephalographic recordings [t.jung, c.humphries, t.lee, s.makeig, m.mckeown, v.iragui, t.sejnowski]

20. reading a neural code [w.bialek, f.rieke, r.steveninck, d.warland]

21. non-boltzmann dynamics in networks of spiking neurons [m.crair, w.bialek]

22. statistical reliability of a blowfly movement-sensitive neuron [r.steveninck, w.bialek]

23. spiking boltzmann machines [g.hinton, a.brown]

24. pulse-firing neural chips for hundreds of neurons [m.brownlow, l.tarassenko, a.murray, a.hamilton, i.han, h.reekie]

25. active noise canceling using analog neuro-chip with on-chip learning capability [j.cho, s.lee]

26. a reconfigurable analog vlsi neural network chip [s.satyanarayana, y.tsividis, h.graf]

others ·
a.smola+b.scholkopf ■
a.barto+s.singh+r.sutton ■
m.jordan ◄
w.bialek ◄
t.sejnowski ◄
g.hinton ◄
p.dayan ◄
current paper ◆

Figure 3: Embedding of 1735 NIPS papers produced by MUHSIC, MVU and PCA. Papers by some representative (combinations of) researchers are highlighted as indicated by the legend. The color bar below each figure shows the eigenspectrum of the learned kernel matrix **K**. The yellow diamond in the graph denotes the current paper as submitted to NIPS. This paper is placed in the location of its nearest neighbor; more details are in the appendix.

i.e. computational neuroscience (which lie on the right). This is be confirmed by examining the highlighted researchers. For instance, the papers by *Smola*, *Schölkopf* and *Jordan* are embedded on the left, whereas the many papers by *Sejnowski*, *Dayan* and *Bialek* can be found on the right.

Unique to the visualization of MUHSIC is that there is a clear grouping of the papers by researchers. For instance, papers on reinforcement learning (*Barto*, *Singh* and *Sutton*) are on the upper left corner; papers by *Hinton* (computational cognitive science) are near the lower left corner; and papers by *Sejnowski* and *Dayan* (computational neuroscientists) are clustered to the right side and adjacent to each other. Interestingly, papers by *Jordan* (at that time best-known for his work in graphical models) are grouped close to the papers on reinforcement learning. This is because *Singh* used to be a postdoc of *Jordan*. Another interesting trend is that papers on new fields of research are embedded on the edges. For instance, papers on reinforcement learning (*Barto*, *Singh* and *Sutton*), are along the left edge. This is consistent with the fact that they presented some interesting new results during this period (recall that the time period of the dataset is 1987 to 1999).

Note that while MUHSIC groups papers according to authors, thereby preserving the macroscopic structure of the data it also reveals the microscopic semantics between the papers. For instance, the 4 papers (numbered from 6 to 9 in Figure 3) by *Smola*, *Scholköpf*, *Hinton* and *Dayan* are very close to each other. Although their titles do not convey strong similarity information, these papers all used handwritten digits for the experiments. A second example are papers by *Dayan*. Although most of his papers are on the neuroscience side, two of his papers (numbered 14 and 15) on reinforcement learning can be found on the machine learning side. A third example are papers by *Bialek* and *Hinton* on spiking neurons (numbered 20, 21 and 23). Although *Hinton's* papers are mainly on the left, his paper on spiking Boltzmann machines is closer to *Bialek's* two papers on spiking neurons.

## 8 Discussion

In summary, MUHSIC provides an embedding of the data which preserves side information possibly available at training time. This way we have a means of controlling *which* representation of the data we obtain rather than having to rely on our luck that the representation found by MVU just happens to match what we want to obtain. It makes feature extraction robust to spurious interactions between observations and noise (see the appendix for an example of adjacency matrices and further discussion). A fortuitous side-effect is that if the matrix containing side information is of low rank, the reduced representation learned by MUHSIC can be lower rank than that obtained by MVU, too. Finally, we showed that MVU and MUHSIC can be formulated as feature extraction for obtaining maximally dependent features. This provides an information theoretic footing for the (brilliant) heuristic of maximizing the trace of a covariance matrix [1].

The notion of extracting features of the data which are maximally dependent on the original data is far more general than what we described in this paper. In particular, one may show that feature selection [7] and clustering [8] can also be seen as special cases of this framework.

**Acknowledgments** NICTA is funded through the Australian Government's *Backing Australia's Ability* initiative, in part through the ARC.This research was supported by the Pascal Network.

## Footnotes

[1]Preprocessed data are available at http://www.it.usyd.edu.au/~lesong/muhsic_datasets.html.

## References

[1] K. Q. Weinberger, F. Sha, and L. K. Saul. Learning a kernel matrix for nonlinear dimensionality reduction. In *Proceedings of the* 21$^{\text{st}}$ *International Conference on Machine Learning*, Banff, Canada, 2004.

[2] J. Sun, S. Boyd, L. Xiao, and P. Diaconis. The fastest mixing markove process on a graph and a connection to a maximum variance unfolding problem. *SIAM Review*, 48(4):681–699, 2006.

[3] A. Gretton, O. Bousquet, A.J. Smola, and B. Schölkopf. Measuring statistical dependence with Hilbert-Schmidt norms. In S. Jain, H. U. Simon, and E. Tomita, editors, *Proceedings Algorithmic Learning Theory*, pages 63–77, Berlin, Germany, 2005. Springer-Verlag.

[4] K. Weinberger, F. Sha, Q. Zhu, and L. Saul. Graph laplacian regularization for large-scale semidefinte programming. In *Neural Information Processing Systems*, 2006.

[5] K. Fukumizu, F. R. Bach, and M. I. Jordan. Dimensionality reduction for supervised learning with reproducing kernel hilbert spaces. *J. Mach. Learn. Res.*, 5:73–99, 2004.

[6] J. Goldberger, S. Roweis, G. Hinton, and R. Salakhutdinov. Neighbourhood component analysis. In *Advances in Neural Information Processing Systems 17*, 2004.

[7] L. Song, A. Smola, A. Gretton, K. Borgwardt, and J. Bedo. Supervised feature selection via dependence estimation. In *Proc. Intl. Conf. Machine Learning*, 2007.

[8] L. Song, A. Smola, A. Gretton, and K. Borgwardt. A dependence maximization view of clustering. In *Proc. Intl. Conf. Machine Learning*, 2007.

